# *Time Dependent Adaptive Neural Networks*

**Fernando J. Pineda**

Center for Microelectronics Technology
Jet Propulsion Laboratory
California Institute of Technology
Pasadena, CA 91109

## ABSTRACT

A comparison of algorithms that minimize error functions to train the trajectories of recurrent networks, reveals how complexity is traded off for causality. These algorithms are also related to time-independent formalisms. It is suggested that causal and scalable algorithms are possible when the activation dynamics of adaptive neurons is fast compared to the behavior to be learned. Standard continuous-time recurrent backpropagation is used in an example.

## 1 INTRODUCTION

Training the time dependent behavior of a neural network model involves the minimization of a function that measures the difference between an actual trajectory and a desired trajectory. The standard method of accomplishing this minimization is to calculate the gradient of an error function with respect to the weights of the system and then to use the gradient in a minimization algorithm (e.g. gradient descent or conjugate gradient).

Techniques for evaluating gradients and performing minimizations are well developed in the field of optimal control and system identification, but are only now being introduced to the neural network community. Not all algorithms that are useful or efficient in control problems are realizable as physical neural networks. In particular, physical neural network algorithms must satisfy locality, scaling and causality constraints. Locality simply is the constraint that one should be able to update each connection using only presynaptic and postsynaptic information. There should be no need to use information from neurons or connections that are not in physical contact with a given connection. Scaling, for this paper, refers to the

scaling law that governs the amount of computation or hardware that is required to perform the weight updates. For neural networks, where the number of weights can become very large, the amount of hardware or computation required to calculate the gradient must scale linearly with the number of weights. Otherwise, large networks are not possible. Finally, learning algorithms must be causal since physical neural networks must evolve forwards in time. Many algorithms for learning time-dependent behavior, although they are seductively elegant and computationally efficient, cannot be implemented as physical systems because the gradient evaluation requires time evolution in two directions. In this paper networks that violate the causality constraint will be referred to as unphysical.

It is useful to understand how scalability and causality trade off in various gradient evaluation algorithms. In the next section three related gradient evaluation algorithms are derived and their scaling and causality properties are compared. The three algorithms demonstrate a natural progression from a causal algorithm that scales poorly to an a causal algorithm that scales linearly.

The difficulties that these exact algorithms exhibit appear to be inescapable. This suggests that approximation schemes that do not calculate exact gradients or that exploit special properties of the tasks to-be-learned may lead to physically realizable neural networks. The final section of this paper suggests an approach that could be exploited in systems where the time scale of the to-be-learned task is much slower than the relaxation time scale of the adaptive neurons.

## 2 ANALYSIS OF ALGORITHMS

We will begin by reviewing the learning algorithms that apply to time-dependent recurrent networks. The control literature generally derives these algorithms by taking a variational approach (e.g. Bryson and Ho, 1975). Here we will take a somewhat unconventional approach and restrict ourselves to the domain of differential equations and their solutions. To begin with, let us take a concrete example. Consider the neural system given by the equation

$$\frac{dx_i}{dt} = x_i + \sum_{i=1}^{n} w_{ij} f(x_j) + I_i \tag{1}$$

Where f(.) is a sigmoid shaped function (e.g. tanh(.)) and $I_i$ is an external input This system is a well studied neural model (e.g. Aplevich, 1968; Cowan, 1967; Hopfield, 1984; Malsburg, 1973; Sejnowski, 1977). The goal is to find the weight matrix w that causes the states x(t) of the output units to follow a specified trajectory x(t). The actually trajectory depends not only on the weight matrix but also on the external input vector **I**. To find the weights one minimizes a measure of the difference between the actual trajectory **x**(t) and the desired trajectory ξ(t). This measure is a functional of the trajectories and a function of the weights. It is given by

$$E(\mathbf{w}, t_f, t_o) = \frac{1}{2} \sum_{i \in O} \int_{t_o}^{t_f} dt \left( x_i(t) - \xi_i(t) \right)^2 \tag{2}$$

where O is the set of output units. We shall, only for the purpose of algorithm comparison,

make the following assumptions: (1) That the networks are fully connected (2) That all the interval $[t_o, t_f]$ is divided into q segments with numerical integrations performed using the Euler method and (3) That all the operations are performed with the same precision. This will allow us to easily estimate the amount of computation and memory required for each algorithm relative to the others.

## 2.1 ALGORITHM A

If the objective function E is differentiated with respect to $w_{rs}$ one obtains

$$\frac{\partial E}{\partial w_{rs}} = -\sum_{i=1}^{n} \int_{t_o}^{t_f} dt \, J_i(t) \, p_{irs}(t) \tag{3a}$$

where

$$J_i = \begin{cases} \xi_i(t) - x_i(t) & \text{if } i \in O \\ 0 & \text{if } i \notin O \end{cases} \tag{3b}$$

and where

$$p_{irs} = \frac{\partial x_i}{\partial w_{rs}} \tag{3c}$$

To evaluate $p_{irs}$, differentiate equation (1) with respect to $w_{rs}$ and observe that the time derivative and the partial derivative with respect to $w_{rs}$ commute. The resulting equation is

$$\frac{dp_{irs}}{dt} = \sum_{j=1}^{n} L_{ij}(x_j) \, p_{jrs} + s_{ir} \tag{4a}$$

where

$$L_{ij}(x_j) = -\delta_{ij} + w_{ij} f'(x_j) \tag{4b}$$

and where

$$s_{irs} = \delta_{ir} f(x_s) \tag{4c}$$

The initial condition for eqn. (4a) is $p(t_o) = 0$. Equations (1), (3) and (4) can be used to calculate the gradient for a learning rule. This is the approach taken by Williams and Zipser (1989) and also discussed by Pearlmutter(1988). Williams and Zipser further observe that one can use the instantaneous value of $p(t)$ and $J(t)$ to update the weights continually provided the weights change slowly. The computationally intensive part of this algorithm occurs in the integration of equation (4a). There are $n^3$ components to $\mathbf{p}$ hence there are $n^3$ equations. Accordingly the amount of hardware or memory required to perform the calculation will scale like $n^3$. Each of these equations requires a summation over all the neurons, hence the amount of computation (measured in multiply-accumulates) goes like $n^4$ per time step, and there are q time steps, hence the total number of multiply-accumulates scales like $n^4 q$   Clearly, the scaling properties of this approach are very poor and it cannot be practically applied to very large networks.

## 2.2 ALGORITHM B

Rather than numerically integrate the system of equations (4a) to obtain $p(t)$, suppose we write down the formal solution. This solution is

$$p_{irs}(t) = \sum_{j=1}^{n} K_{ij}(t,t_o) \, p_{jrs}(t_o) + \sum_{j=1}^{n} \int_{t_o}^{t} dt K_{ij}(t,\tau) \, s_{jrs}(\tau) \tag{5a}$$

The matrix $\mathbf{K}$ is defined by the expression

$$K(t_2, t_1) = exp\left( \int_{t_1}^{t_2} d\tau \, L(x(\tau)) \right) \tag{5b}$$

This matrix is known as the propagator or transition matrix. The expression for $p_{irs}$ consists of a homogeneous solution and a particular solution. The choice of initial condition $p_{irs}(t_o)$ = 0 leaves only the particular solution. If the particular solution is substituted back into eqn. (3a), one eventually obtains the following expression for the gradient

$$\frac{\partial E}{\partial w_{rs}} = -\sum_{i=1}^{n} \int_{t_o}^{t_f} dt \int_{t_o}^{t} d\tau \, J_i(t) K_{ir}(t,\tau) f(x_s(\tau)) \tag{6}$$

To obtain this expression one must observe that $s_{jrs}$ can be expressed in terms of $x_s$, i.e. use eqn. (4c). This allows the summation over j to be performed trivially, thus resulting in eqn.(6). The familiar outer product form of backpropagation is not yet manifest in this expression. To uncover it, change the order of the integrations. This requires some care because the limits of the integration are not the same. The result is

$$\frac{\partial E}{\partial w_{rs}} = -\sum_{i=1}^{n} \int_{t_o}^{t_f} d\tau \int_{\tau}^{t_f} dt \, J_i(t) K_{ir}(t,\tau) f(x_s(\tau)) \tag{7}$$

Inspection of this expression reveals that neither the summation over i nor the integration over $\tau$ includes $x_s(t)$, thus it is useful to factor it out. Consequently equation (7) takes on the familiar outer product form of backpropagation

$$\frac{\partial E}{\partial w_{rs}} = -\int_{\tau}^{t_f} dt \, y_r(t) f(x_s(t)) \tag{8}$$

Where $y_r(t)$ is defined to be

$$y_r(\tau) = -\sum_{i=1}^{n} \int_{\tau}^{t_f} dt \, J_i(t) K_{ir}(t,\tau) \tag{9}$$

Equation (8), defines an expression for the gradient, provided we can calculate $y_r(t)$ from eqn. (9). In principle, this can be done since the propagator $\mathbf{K}$ and the vector $\mathbf{J}$ are both completely determined by $\mathbf{x}(t)$. The computationally intensive part of this algorithm is the calculation of $\mathbf{K}(t,\tau)$ for all values of t and $\tau$. The calculation requires the integration of equations of the form

$$\frac{dK(t,\tau)}{dt} = L(x(t)) K(t,\tau) \tag{10}$$

for q different values of $\tau$. There are $n^2$ different equations to integrate for each value of $\tau$ Consequently there are $n^2q$ integrations to be performed where the interval from $t_o$ to $t_f$ is divided into q intervals. The calculation of all the components of $\mathbf{K}(t,\tau)$, from $t_f$ to $t_o$, scales like $n^3q^2$, since each integration requires n multiply-accumulates per time step and there are q time steps. Similarly, the memory requirements scale like $n^2q^2$. This is because $\mathbf{K}$ has $n^2$ components for each $(t,\tau)$ pair and there are $q^2$ such pairs.

Equation (10) must be integrated forwards in time from t= τ to t = t,*and backwards in time* from t= τ to t = t₀. This is because **K** must satisfy **K**( τ,τ) = **1** (the identity matrix) for all τ. This condition follows from the definition of **K** eqn. (5b). Finally, we observe that expression (9) is the time-dependent analog of the expression used by Rohwer and Forrest (1987) to calculate the gradient in recurrent networks. The analogy can be made somewhat more explicit by writing $\mathbf{K}(t,\tau)$ as the inverse $\mathbf{K}^{-1}(\tau,t)$. Thus we see that **y**( t ) can be expressed in terms of a matrix inverse just as in the Rohwer and Forrest algorithm.

## 2.3     ALGORITHM C

The final algorithm is familiar from continuous time optimal control and identification. The algorithm is usually derived by performing a variation on the functional given by eqn. (2). This results in a two-point boundary value problem. On the other hand, we know that **y** is given by eqn. (9). So we simply observe that this is the particular solution of the differential equation

$$-\frac{dy}{dt} = L^{T}(x(t))y + J \tag{11}$$

Where **L**ᵀ is the transpose of the matrix defined in eqn. (4b). To see this simply substitute the form for **y** into eqn. (11) and verify that it is indeed the solution to the equation.

The particular solution to eqn. (11) vanishes only if $y(t_f) = 0$. In other words: to obtain y(t) we need only integrate eqn. (11) *backwards* from the final condition $y(t_f) = 0$. This is just the algorithm introduced to the neural network community by Pearlmutter (1988). This also corresponds to the unfolding in time approach discussed by Rumelhart et al. (1986), provided that all the equations are discretized and one takes Δt = 1.

The two point boundary value problem is rather straight forward to solve because the equation for x(t) is independent of y(t). Both x(t) and y(t) can be obtained with n multiply-accumulates per time step. There are q time steps from t₀ to t_f and both x(t) and y(t) have n components, hence the calculation of x(t) and y(t) scales like n²q. The weight update equation also requires n²q multiply- accumulates. Thus the computational requirements of the algorithm as a whole scale like n²q The memory required also scales like n²q , since it is necessary to save each value of x(t) along the trajectory to compute y(t).

## 2.4     SCALING VS CAUSALITY

The results of the previous sections are summarized in table 1 below. We see that we have a progression of tradeoffs between scaling and causality. That is, we must choose between a causal algorithm with exploding computational and storage requirements and an a causal algorithm with modest storage requirements. There is no q dependence in the memory requirments because the integral given in eqn. (3a) can be accumulated at each time step. Algorithm B has some of the worst features of both algorithms.

**Table 1:** Comparison of three algorithms

| Algorithm | Memory | Multiply -accumulates | diirection of integations |
|---|---|---|---|
| A | $n^3$ | $n^4q$ | **x** and **p** are both forward in time |
| B | $n^2q^2$ | $n^3q^2$ | **x** is forward, **K** is forward *and* backward |
| C | $n^2q$ | $n^2q$ | **x** is forward, **y** is backward in time. |

Digital hardware has no difficulties (at least over finite time intervals) with a causal algorithms provided a stack is available to act as a memory that can recall states in reverse order. To the extent that the gradient calculations are carried out on digital machines, it makes sense to use algorithm C because it is the most efficient. In analog VLSI however, it is difficult to imagine how to build a continually running network that uses an a causal algorithm. Algorithm A is attractive for physical implementation because it could be run continually and in real time (Williams and Zipser, 1989). However, its scaling properties preclude the possibility of building very large networks based on the algorithm. Recently, Zipser (1990) has suggested that a divide and conquer approach may reduce the computational and spatial complexity of the algorithm. This approach, although promising, does not always work and there is as yet no convergence proof. How then, is it possible to learn trajectories using local, scalable and causal algorithms? In the next section a possible avenue of attack is suggested.

## 3 EXPLOITING DISPARATE TIME SCALES

I assert that for some classes of problems there are scalable and causal algorithms that approximate the gradient and that these algorithms can be found by exploiting the disparity in time scales found in these classes of problems. In particular, I assert that when the time scale of the adaptive units is fast compared to the time scale of the behavior to be learned, it is possible to find scalable and causal adaptive algorithms. A general formalism for doing this will not be presented here, instead a simple, perhaps artificial, example will be presented. This example minimizes an error function for a time dependent problem.

It is likely that trajectory generation in motor control problems are of this type. The characteristic time scales of the trajectories that need to be generated are determined by inertia and friction. These mechanical time scales are considerably longer than the electronic time scales that occur in VLSI. Thus it seems that for robotic problems, there may be no need to use the completely general algorithms discussed in section 2. Instead, algorithms that take advantage of the disparity between the mechanical and the electronic time scales are likely to be more useful for learning to generate trajectories.

he task is to map from a periodic input $I(t)$ to a periodic output $\xi(t)$. The basic idea is to use the continuous-time recurrent-backpropagation approach with slowly varying time-dependent inputs rather than with static inputs. The learning is done in real-time and in a continuous fashion. Consider a set of n "fast" neurons (i=1,..,n) each of which satisfies the

additive activation dynamics determined by eqn (1).  Assume that the initial weights are sufficiently small that the dynamics of the network would be convergent *if the inputs* **I** *were constant*.  The external input vector $\xi$ is applied to the network through the vector **I**.  It has been previously shown (Pineda, 1988) that the ij-th component of the gradient of E is equal to $y_i^f f(x_j^f)$ where $x_j^f$ is the steady state solution of eqn. (1) and where $y_i^f$ is a component of the steady state solution of

$$\frac{dy}{dt} = \mathbf{L}^T(\mathbf{x}^f)y + \mathbf{J} \tag{12}$$

where the components of $\mathbf{L}^T$ are given by eqn. (4.b).  Note that the relative sign between equations (11) and (12) is what enables this algorithm to be causal.  Now suppose that instead of a fixed input vector **I**, we use a slowly varying input $\mathbf{I}(t/\tau_p)$ where $\tau_p$ is the characteristic time scale over which the input changes significantly. If we take as the gradient descent algorithm, the dynamics defined by

$$\tau_w \frac{dw_{rs}}{dt} = y_i(t)x_j(t) \tag{13}$$

where $\tau_w$ is the time constant that defines the (slow) time scale over which w changes and where $x_j$ is the *instantaneous* solution of eqn. (1) and $y_i$ is the *instantaneous* solution of eqn.(12) . Then in the adiabatic limit the Cartesian product $y_i f(x_j)$ in eqn. (13) approximates the negative gradient of the objective function E, that is

$$y_r^f f(x_s^f(t)) \cong y_r(t)f(x_s(t)) \tag{14}$$

This approach can map one continuous trajectory into another continuous trajectory, provided the trajectories change slowly enough.  Furthermore, learning occurs causally and scalably.   There is no memory in the model, i.e. the output of the adaptive neurons depends only on their input and not on their internal state.  Thus, this network can never learn to perform tasks that require memory unless the learning algorithm is modified to learn the appropriate transitions.   This is the major drawback of the adiabatic approach.  Some state information can be incorporated into this model by using recurrent connections — in which case the network can have multiple basins and the final state will depend on the initial state of the net as well as on the inputs, but this will not be pursued here.

Simple simulations were performed to verify that the approach did indeed perform gradient descent.  One simulation is presented here for the benefit of  investigators who may wish to verify the results. A feedforward network topology consisting of two input units, five hidden units and two output units was used for the adaptive network.  Units were numbered sequentially, 1 through 9, beginning with the input layer and ending in the output layer.  Time dependent external inputs for the two  input  neurons were generated with time dependence $I_1 = \sin(2\pi t)$ and $I_2 = \cos(2\pi t)$.  The targets for the output neurons were $\xi_8 = R \sin(2\pi t)$ and $\xi_9 = R \cos(2\pi t)$ where $R = 1.0 + 0.1\sin(6\pi t)$. All the equations were simultaneously integrated using 4th order Runge-Kutta with a time step of 0.1.  A relaxation time scale was introduced into the forward and backward propagation equations by multiplying the time derivatives in eqns. (1) and (12) by $\tau_x$ and $\tau_y$ respectively.  These time scales were set to $\tau_x = \tau_y = 0.5$.  The adaptive time scale of the weights was $\tau_w = 1.0$.  The error in the network was initially, E =

10 and the integration was cut off when the error reached a plateau at $E = 0.12$. The learning curve is shown in Fig. 1. The trained trajectory did not exactly reach the desired solution. In particular the network did not learn the odd order harmonic that modulates R. By way of comparison, a conventional backpropagation approach that calculated a cumulative gradient over the trajectory and used conjugate gradient for the descent, was able to converge to the global minimum.

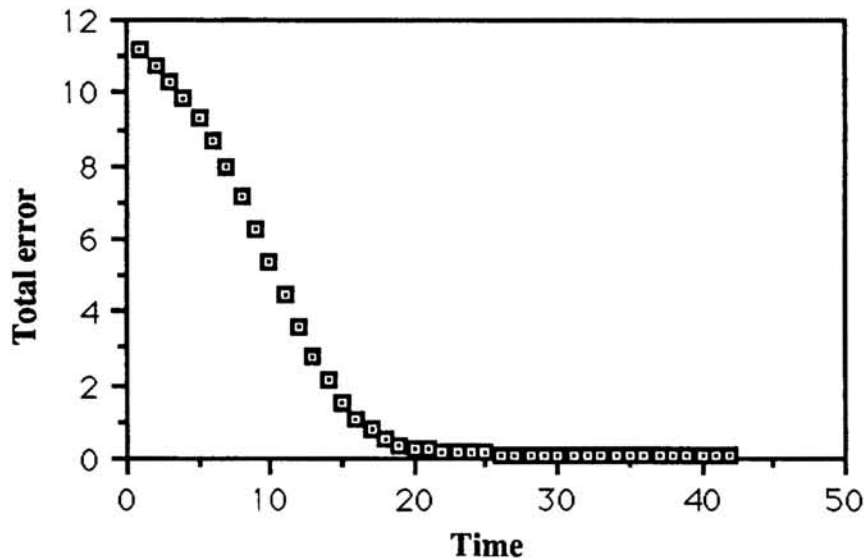

**Figure 1:** Learning curve. One time unit corresponds to a single oscillation

## 4 SUMMARY

The key points of this paper are: 1) Exact minimization algorithms for learning time-dependent behavior either scale poorly or else violate causality and 2) Approximate gradient calculations will likely lead to causal and scalable learning algorithms. The adiabatic approach should be useful for learning to generate trajectories of the kind encountered when learning motor skills.

References herein to any specific commercial product, process, or service by trade name, trademark, manufacturer, or otherwise, does not constitute or imply any endorsement by the United States Government or the Jet Propulsion Laboratory, California Institute of Technology. The work described in this paper was carried out at the Center for Space Microelectonrics Technology, Jet Propulsion Laboratory, California Institute of Technology. Support for the work came from the Air Force Office of Scientific Research through an agreement with the National Aeronautics and Space Administration (AFOSR-ISSA-90-0027).

## REFERENCES

Aplevich, J. D. (1968). Models of certain nonlinear systems. In E. R. Caianiello (Ed.), *Neural Networks*, (pp. 110-115). Berlin: Springer Verlag.

Bryson, A. E. and Ho, Y. (1975). *Applied Optimal Control: Optimization, Estimation, and*

*Control.* New York: Hemisphere Publishing Co.

Cowan, J. D. (1967). A mathematical theory of central nervous activity. Unpublished dissertation, Imperial College, University of London.

Hopfield, J. J. (1984). Neurons with graded response have collective computational properties like those of two-state neurons. *Proc. Nat. Acad. Sci. USA, Bio.*, 81, 3088-3092.

Malsburg, C. van der (1973). Self-organization of orientation sensitive cells in striate cortex, *Kybernetic*, 14, 85-100.

Pearlmutter, B. A. (1988), Learning state space trajectories in recurrent neural networks: A preliminary report, *(Tech. Rep. AIP-54)*, Department of Computer Science, Carnegie Mellon University, Pittsburgh, PA

Pineda, F. J. (1988). Dynamics and Architecture for Neural Computation. *Journal of Complexity*, 4, (pp.216-245)

Rowher R, R. and Forrest, B. (1987). Training time dependence in neural networks, In M. Caudil and C. Butler, (Eds.), *Proceedings of the IEEE First Annual International Conference on Neural Networks*, 3, (pp. 701-708). San Diego, California: IEEE.

Rumelhart, D. E., Hinton, G. E., and Willaims, R.J. (1986).   Learning Internal Representations by Error Propagation. In D. E. Rumelhart and J. L. McClelland, (Eds.), *Parallel Distributed Processing*, (pp. 318-362). Cambridge: M.I.T. Press.

Sejnowski, T. J. (1977). Storing covariance with nonlinearly interacting neurons. *Journal of Mathematical Biology*, 4, 303-321.

Williams, R.J. and Zipser, D. (1989). A learning algorithm for continually running fully recurrent neural networks. *Neural Computation*, 1, (pp. 270-280).

Zipser, D. (1990). Subgrouping reduces complexity and speeds up learning in recurrent networks, (this volume).
